# Dynamically Adaptable CMOS Winner-Take-All Neural Network

**Kunihiko Iizuka, Masayuki Miyamoto and Hirofumi Matsui**
Information Technology Research Laboratories
Sharp
Tenri, Nara, JAPAN

## Abstract

The major problem that has prevented practical application of analog neuro-LSIs has been poor accuracy due to fluctuating analog device characteristics inherent in each device as a result of manufacturing. This paper proposes a dynamic control architecture that allows analog silicon neural networks to compensate for the fluctuating device characteristics and adapt to a change in input DC level. We have applied this architecture to compensate for input offset voltages of an analog CMOS WTA (Winner-Take-All) chip that we have fabricated. Experimental data show the effectiveness of the architecture.

## 1   INTRODUCTION

Analog VLSI implementation of neural networks, such as silicon retinas and adaptive filters, has been the focus of much active research. Since it utilizes physical laws that electric devices obey for neural operation, circuit scale can be much smaller than that of a digital counterpart and massively parallel implementation is possible. The major problem that has prevented practical applications of these LSIs has been fluctuating analog device characteristics inherent in each device as a result of manufacturing. Historically, this has been the main reason most analog devices have been superseded by digital devices. Analog neuro VLSI is expected to conquer this problem by making use of its adaptability. This optimistic view comes from the fact that in spite of the unevenness of their components, biological neural networks show excellent competence.

This paper proposes a CMOS circuit architecture that dynamically compensates for fluctuating component characteristics and at the same time adapts device state to incoming signal levels. There are some engineering techniques available to compensate

for MOS threshold fluctuation, e.g., the chopper comparator, but they need a periodical change of mode to achieve the desired effect. This is because there are two modes one for the adaptation and one for the signal processing. This is quite inconvenient because extra clock signals are needed and a break of signal processing takes place.

Incoming signals usually consist of a rapidly changing foreground component and a slowly varying background component. To process these signals incessantly, biological neural networks make use of multiple channels having different temporal/spatial scales. While a relatively slow/large channel is used to suppress background floating, a faster/smaller channel is devoted to process the foreground signal. The proposed method inspired by this biological consideration utilizes different frequency bands for adaptation and signal processing (Figure 1), where negative feedback is applied through a low pass filter so that the feedback will not affect the foreground signal processing.

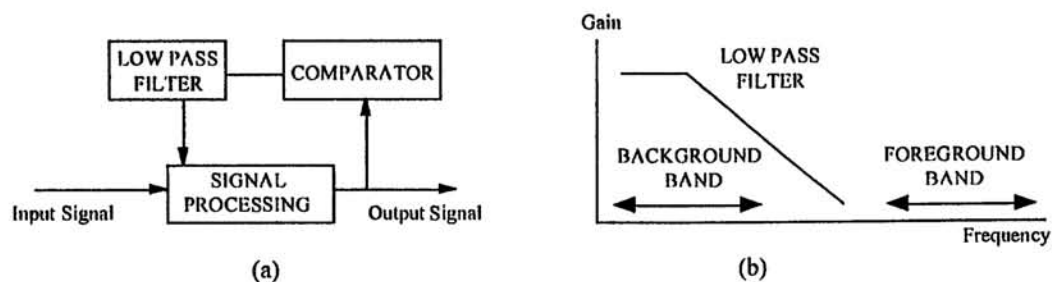

Figure 1: Dynamic adaptation by frequency divided control. (a) model diagram, (b) frequency division.

In the first part of this paper, a working analog CMOS WTA chip that we have test fabricated is introduced. Then, dynamical adaptation for this WTA chip is described and experimental results are presented.

## 2   ANALOG CMOS WTA CHIP

### 2.1   ARCHITECTURE AND SPECIFICATION

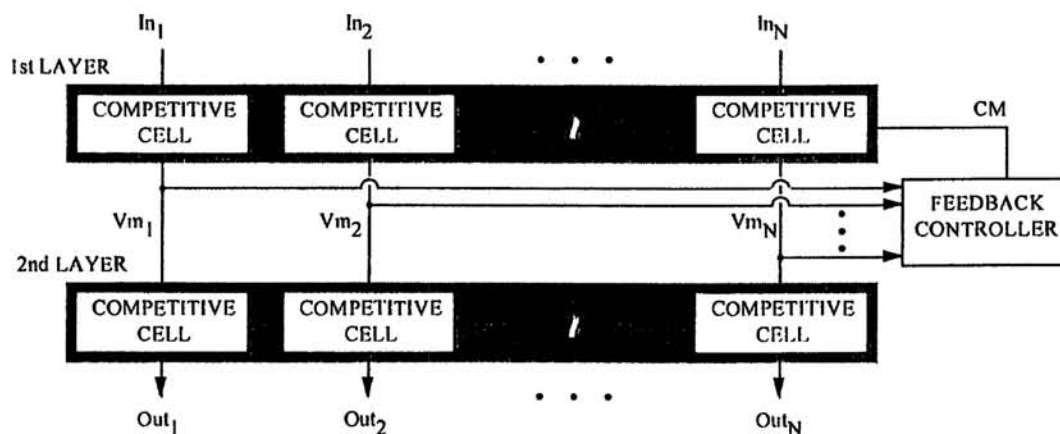

Figure 2: Analog CMOS WTA chip architecture

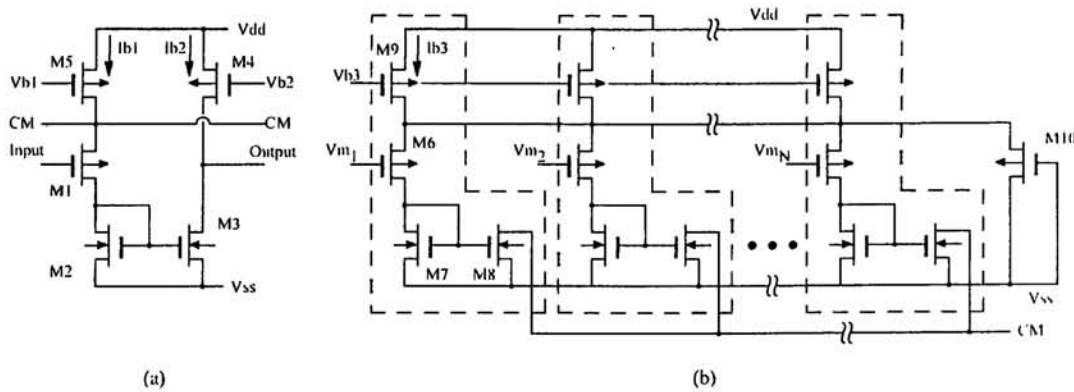

Figure 3: Circuit diagrams for (a) the competitive cell and (b) the feedback controller.

As a basic building block to construct neuro-chips, analog WTA circuits have been investigated by researchers such as [Lazzaro, 1989] and [Pedroni, 1994]. All CMOS analog WTA circuits are based on voltage follower circuits [Pedroni, 1995] to realize competition through inhibitory interaction, and they use feedback mechanisms to enhance resolution gain. The architecture of the chip that we have fabricated is shown in Figure 2 and the circuit diagram is in Figure 3. This WTA chip indicates the lowest input voltage by making the output voltage corresponds to the lowest input voltage near Vss (winner), and others nearly the power supply voltage Vdd (loser). The circuit is similar to [Sheu, 1993], but represents two advances.

1. The steering current that the feedback controller absorbs from the line CM is enlarged, allowing the winner cell can compete with others in the region where resolution gain is the largest.

2 The feedback controller originally placed after the second competitive layer is removed in order to guarantee the existence of at least one output node whose voltage is nearly zero.

Table 1 shows the specifications of the fabricated chip.

Table 1: Specifications of the fabricated WTA chip

| Process | 0.8 um double-metal CMOS |
|---|---|
| Number of input nodes | 32 |
| Power dissipation (measured) | < 480 uW |
| Power supply voltage | 3V |
| Resolution (theoretical) | 10 mV |
| Settling time (measured) | 5 usec |
| Die area | 1 mm × 0.5 mm |

## 2.2  INPUT OFFSET VOLTAGE

Input offset voltages of a WTA chip may greatly deteriorate chip performance. Examples of input offset voltage distribution of the fabricated chips are shown in Figure 4. Each input offset voltage is measured relative to the first input node. The input offset voltage

$\Delta V_j$ of the j-th input node is defined as $\Delta V_j = Vin_j - Vin_l$ when the voltages of output nodes $Out_j$ and $Out_l$ are equal; $Vin_l$ is fixed to a certain voltage and the voltage of other input nodes are fixed at a relatively high voltage.

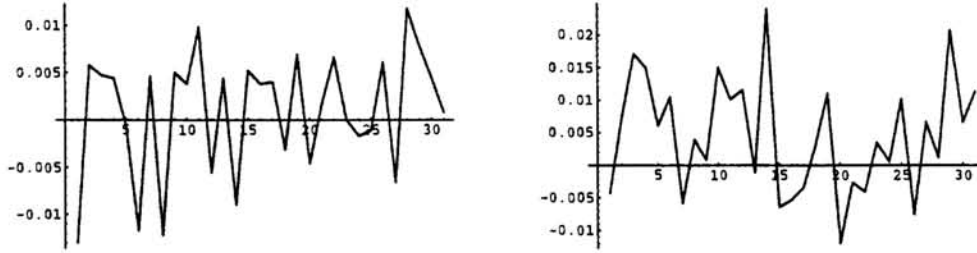

Figure 4: Examples of measured input offset voltage distribution.

The primary factor of the input offset voltage is considered to be fluctuation of MOS transistor threshold voltages in the first layer competitive cell. Then, the input offset voltage $\Delta V_j$ of this cell yielded by the small fluctuation $\Delta Vth^i$ of $Vth^i$ is calculated as follows:

$$\Delta V_j \approx \sum_{i=1}^{4} \Delta Vth^i \frac{\partial Vin}{\partial Vth^i}$$

$$= -\Delta Vth^1 + \frac{gd_1 + gd_2 + gm_2}{gm_1}(\Delta Vth^2 - \Delta Vth^3) + \frac{gm_4(gd_1 + gd_2 + gm_2)}{gm_1 gm_3}\Delta Vth^4 \quad ,$$

where $gm_i$ and $gd_i$ are the transconductance and the drain conductance of MOS Mi, respectively. Using design and process parameters, we can estimate the input offset voltage to be

$$\Delta V_j \approx -\Delta Vth^1 + (\Delta Vth^2 - \Delta Vth^3) + 0.15\Delta Vth^4 \quad ,$$

Based on our experiences, the maximum fluctuation of $Vth^i$ in a chip is usually smaller than 20 mV, and it is reasonable to consider that the difference $|\Delta Vth^2 - \Delta Vth^3|$ is even smaller; perhaps less than 5 mV, because M2 and M3 compose a current mirror and are closely placed. This implies that the maximum of $\Delta V_j$ is about 28 mV, which is in rough agreement with the measured data.

## 3   DYNAMICAL ADAPTATION ARCHITECTURE

In Figure 5, we show circuit implementation of the dynamically adaptable WTA function. In each feedback channel, the difference between each output and the reference $Vref$ is fed back to the input node through a low pass filter consisting of R and C. The charge stored in capacitor C is controlled by this feedback signal.

Let the linear approximation of the WTA chip DC characteristic be

$$Vout_i = A ( Vin_i - V0_i),$$

where $Vin_i$ and $Vout_i$ are the voltages at the nodes $In_i$ and $Out_i$ respectively, and $A$ and $V0_i$ are functions of $Vin_j$ ( $j \neq i$ ). The input offset voltage relative to the node $In_l$ is considered to be the difference between $V0_i$ and $V0_l$. On the other hand, the DC characteristic of the i-th feedback path can be approximated as

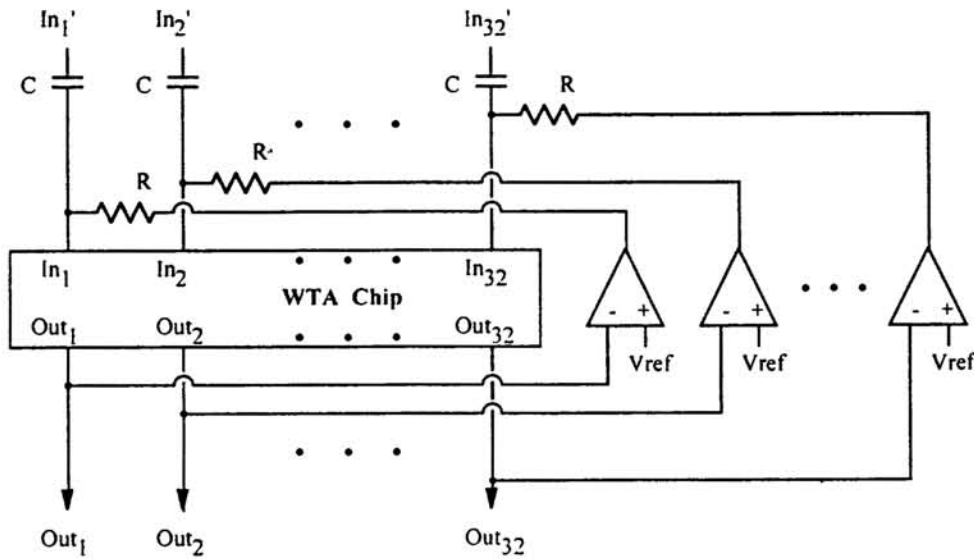

Figure 5: WTA chip equipped with adaptation circuit where R=10MΩ and C=0.33μF.

$$Vin_i = B \ (\ Vout_i - Vref).$$

It follows from the above two equations that

$$Vin_i = -\frac{AB}{1-AB} \ VO_i - \frac{B}{1-AB} \ Vref \approx VO_i \ .$$

The last term is derived using the assumptions $A \gg 1$ and $B \ll -1$. This means that the voltage difference between the DC level of the input and $VO_i$ is clamped on the capacitor C. This in turn implies that the input offset voltage will be successfully compensated for.

The role of the low pass filters is twofold.

1. They guarantee stable dynamics of the feedback loop; we can make the cutoff frequency of the low pass filters small enough so that the gain of the feedback path is attenuated before the phase of the feedback signal is delayed by more than 180° .

2. They prevent the feed-forward WTA operation from being affected, as shown in Figure 1, the adaptive control is carried out on a different, non-overlapped frequency band than WTA operation.

## 4  EXPERIMENTAL RESULTS

Experiments concerning the adaptable WTA function were carried out by applying pulses of 90% duty to the input nodes $In'_1$ and $In'_2$, while other input nodes were fixed to a certain voltage. In Figures 6 (a) and 6 (b), the output waveforms of $Out_1$, $Out_2$, $Out_3$ and the waveform of the pulse applied to the node $In'_1$ are shown. Figure 6(a) shows the result when the same pulse was applied to both $In'_1$ and $In'_2$. Figure 6(b) shows the result when the amplitude of the pulse to $In'_1$ was greater than that of the pulse to $In'_2$ by 10 mV. The schematic explanation of this behavior is in Figure 7. The outputs remained at the same levels for a while after the inputs were shut off, since there was no strong inducement. As a result of adaptation, the winning frequencies of every output nodes become equal in a long time scale. This explains the unstable output during the period of quiescent inputs.

The chip used in this measurement had a relative input offset voltage of 15 mV between nodes $In_1$ and $In_2$. We can see in Figure 6 (a) that this offset voltage was completely compensated for because the output waveforms of corresponding nodes were the same.

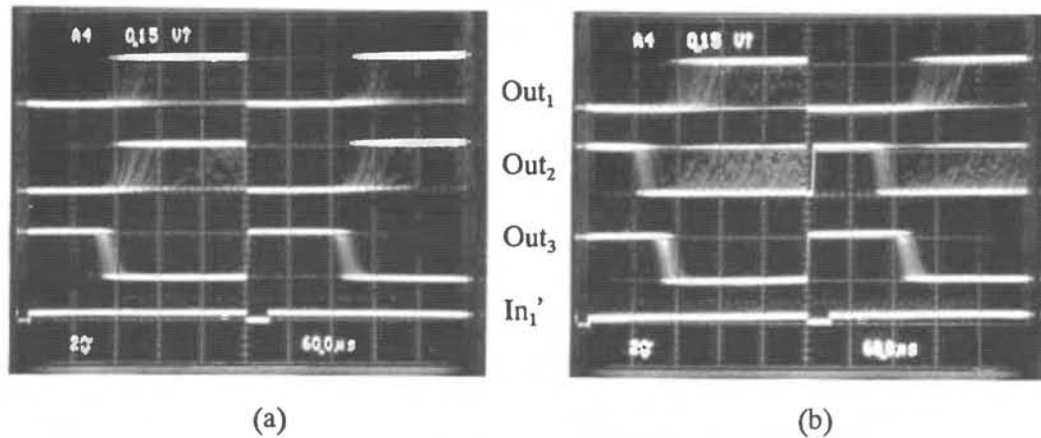

(a)                                                    (b)

Figure 6: The output waveforms ot the dynamically adaptable CMOS WTA neural network. Pulse waves were applied to nodes $In'_1$ and $In'_2$; other nodes voltages were fixed. When the amplitude of each pulse was the same (a), the corresponding output waveforms were the same. When the amplitude of the pulse fed to $In'_1$ was greater than that to $In'_2$ by 10 mV (b), the output voltage at $Out_1$ was low (winner) and that at $Out_2$ was high (loser) during the period the pulse was low (on).

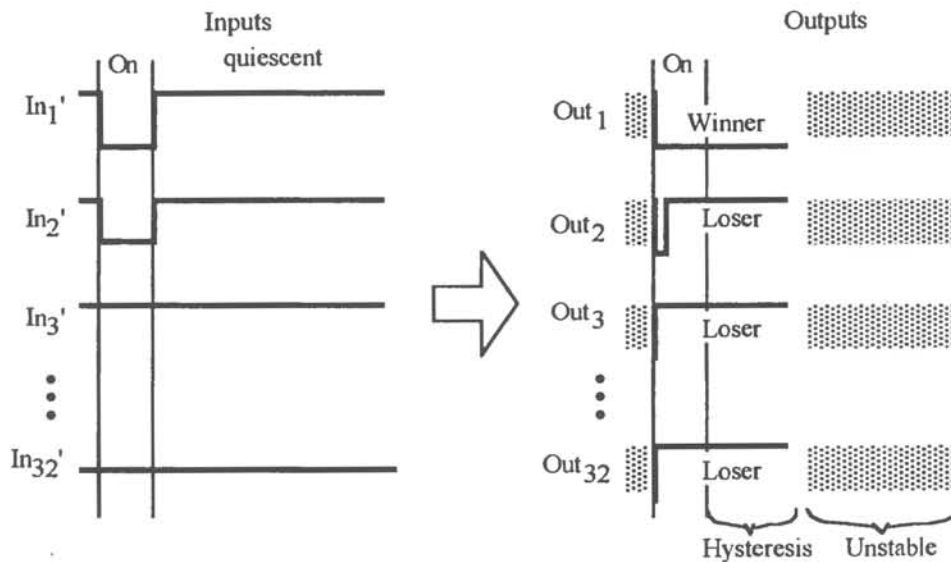

Figure 7: The schematic explanation of the dynamically adaptable WTA behavior.

## 5  CONCLUSION

We have proposed a dynamic adaptation architecture that uses frequency divided control and applied this to a CMOS WTA chip that we have fabricated. Experimental results show that the architecture successfully compensated for input offset voltages of the WTA

chip due to inherent device characteristic fluctuations. Moreover, this architecture gives analog neuro-chips the ability to adapt to incoming signal background levels. This adaptability has a lot of applications. For example, in vision chips, the adaptation may be used to compensate for the fluctuation of photo sensor characteristics, to adapt the gain of photo sensors to background illumination level and to automatically control color balance. As another application, Figure 8 describes an analog neuron with weighted synapses, where the time constant RC is much larger than the time constant of input signals.

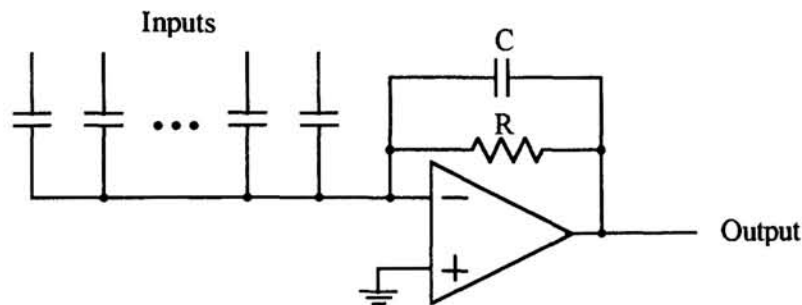

Figure 8: Analog neuron with weighted synapses where the time constant RC is much larger than that of input signals.

The key to this architecture is use of non-overlapping frequency bands for adaptation to background and foreground signal processing. For neuro-VLSIs, this requires implementing circuits with completely different time scale constants. In modern VLSI technology, however, this is not a difficult problem because processes for very high resistances, i.e., teraohms, are available.

## Acknowledgment

The authors would like to thank Morio Osaka for his help in chip fabrication and Kazuo Hashiguchi for his support in experimental work.

## References

Choi, J. & Sheu, B.J. (1993) A high-precision VLSI winner-take-all circuit for self-organizing neural networks. *IEEE J. Solid-State Circuits*, vol.28, no.5, pp.576-584.

Lazzaro, J., Ryckebush, S., Mahowald, M.A., & Mead, C. (1989) Winner-take-all networks of O(N) complexity. In D.S. Touretzky (eds.), *Advances in Neural Information Processing Systems* 1, pp. 703-711. Cambridge, MA: MIT Press.

Pedroni, V.A. (1994) Neural n-port voltage comparator network, Electron. Lett., vol.30, no.21, pp1774-1775.

Pedroni, V.A. (1995) Inhibitory Mechanism Analysis of Complexity O(N) MOS Winner-Take-All Networks. *IEEE Trans. Circuits Syst. I*, vol.42, no.3, pp.172-175.